# How to Combine Color and Shape Information for 3D Object Recognition: Kernels do the Trick

**B. Caputo**
Smith-Kettlewell Eye Research Institute,
2318 Fillmore Street,
94115 San Francisco, California, USA
*caputo@ski.org*

**Gy. Dorkó**
Department of Computer Science,
Chair for Pattern Recognition,
University of Erlangen-Nuremberg,
*dorko@informatik.uni-erlangen.de*

## Abstract

This paper presents a kernel method that allows to combine color and shape information for appearance-based object recognition. It doesn't require to define a new common representation, but use the power of kernels to combine different representations together in an effective manner. These results are achieved using results of statistical mechanics of spin glasses combined with Markov random fields via kernel functions. Experiments show an increase in recognition rate up to 5.92% with respect to conventional strategies.

## 1 Introduction

Consider the two cars in Figure 1. They look very similar, but this wouldn't be the case if we would look at color pictures: as the left car is yellow and the right car is red, we would realize at a first glance that they are different. This simple example shows that color and shape information are both important cues for object recognition. In spite of this, just a few systems employ both. This is because most of representations proposed in literature aren't suitable for both type of information [5, 11, 13, 2]. Some authors tackled this problem building up new representations, containing both color and shape information; these approaches show very good performances [7, 12, 6]. However, this strategy has two important drawbacks:

• **both types of information must be used always.**

Although there are many cases where it is convenient to have both, a huge literature shows that color only, or shape only representations work very well for many applications [9, 13, 11, 2]. A new, common representation doesn't always permit to use just color or just shape information alone, depending on the task considered;

• **the dimension of the feature vector.**

If the new representation brings as much information as separate representations do, then we must expect it to have a higher dimensionality than each separate

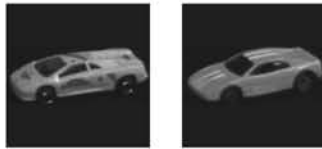

Figure 1: An example of objects similar with respect to shape but not with respect to color (the left car is yellow while the right car is red).

representation alone, with all the risks of a curse of dimensionality effect. If the dimension of the new representation vector is kept under control, we can expect that the representation contains less information that single ones, with a possible decrease of effectiveness

Our goal in this paper is to present a system that uses both types of information while keeping them distinct, allowing the flexibility to use the information sometimes combined, sometimes separated, depending on the application considered. We achieve this goal focusing the attention on how two given shape and color representations can be combined together as they are, rather than define a new representation. We obtain this using Spin Glass-Markov Random Fields (SG-MRF), a new kernel method that integrates results of statistical physics of spin glasses with Gibbs probability distributions via nonlinear kernel mapping. SG-MRFs have been used for robust appearance-based object recognition with very good results, using a kernelized Hopfield energy [3]. Here we extend SG-MRF to a new SG-like energy function, inspired by the ultrametric properties of the SG phase space. The structure of this energy provides a natural framework for combining shape and color representations together, without defining a new common representation (such as a concatenated one, see for instance [7]). This approach presents two main advantages:

- it permits us to use existing and well tested representations both for shape and color information;

- it permits us to use this knowledge in a flexible manner, depending on the task considered.

To the best of our knowledge, there are no previous similar approaches to this problem. Experimental results show the effectiveness of the new proposed kernel method. The paper is organized as follows: section 2 defines the probabilistic framework for object recognition, section 3 reviews SG-MRF and section 4 presents the new energy function and how it can be used for combining together color and shape information. Section 5 presents experiments that show the effectiveness of our approach, compared to other conventional strategies (NNC, $\chi^2$ and SVM [10, 14]). The paper concludes with a summary discussion.

## 2 Probabilistic Appearance-based Object Recognition

Probabilistic appearance-based object recognition methods consider images as random feature vectors. Let $\mathbf{x} \equiv [x_{ij}], i = 1, \ldots \mathcal{N}, j = 1, \ldots \mathcal{M}$ be an $\mathcal{M} \times \mathcal{N}$ image. We will consider each image as a random feature vector $\mathbf{x} \in \Re^{\mathcal{M}\mathcal{N}}$. Assume we have $k$ different classes $\Omega_1, \Omega_2, \ldots, \Omega_k$ of objects, and that for each object is

given a set of $n_j$ data samples, $d_j = \{\mathbf{x}_1^j, \mathbf{x}_2^j, \ldots, \mathbf{x}_{n_j}^j\}, j = 1, \ldots k$. We will assign each object to a pattern class $\Omega_1, \Omega_2, \ldots, \Omega_k$. How the object class $\Omega_j$ is represented, given a set of data samples $d_j$ (relative to that object class), varies for different appearance-based approaches: it can consider shape information only, or color information only or both. This is equivalent to consider a set of features $\{\mathbf{h}_1^j, \mathbf{h}_2^j, \ldots, \mathbf{h}_{n_j}^j\}, j = 1, \ldots k$, where each feature vector $\mathbf{h}_{n_j}^j$ is computed from the image $\mathbf{x}_{n_j}^j$, $\mathbf{h}_{n_j}^j = T(\mathbf{x}_{n_j}^j), \mathbf{h}_{n_j}^j \in G \equiv \Re^m$. Assuming that the data samples $d_j$ are a sufficient statistic for the pattern class $\Omega_j$, the goal will be to estimate the probability distribution $P_{\Omega_j}(\mathbf{h})$ that has generated them. Then, given a test image $\mathbf{x}$ and its associate feature vector $\mathbf{h}$, the decision will be made using a Maximum A Posteriori (MAP) classifier:

$$j^* = \operatorname*{argmax}_j P_{\Omega_j}(\mathbf{h}) = \operatorname*{argmax}_j P(\Omega_j|\mathbf{h}) = \operatorname*{argmax}_j P(\mathbf{h}|\Omega_j)P(\Omega_j), \qquad (1)$$

using Bayes rule. $P(\mathbf{h}|\Omega_j)$ are the Likelihood Functions (LFs) and $P(\Omega_j)$ are the prior probabilities of the classes. In the rest of the paper we will assume that the prior $P(\Omega_j)$ is the same for all object classes; thus the Bayes classifier (1) simplifies to

$$j^* = \operatorname*{argmax}_j P(\mathbf{h}|\Omega_j). \qquad (2)$$

A possible strategy for modeling $P(\mathbf{h}|\Omega_j)$ is to use Gibbs distributions within a Markov Random Field (MRF) framework. The MRF joint probability distribution is given by

$$P(\mathbf{h}|\Omega_j) = \frac{1}{Z} \exp\left(-E(\mathbf{h}|\Omega_j)\right), \qquad Z = \sum_{\{\mathbf{h}\}} \exp\left(-E(\mathbf{h}|\Omega_j)\right). \qquad (3)$$

The normalizing constant $Z$ is called the partition function, and $E(\mathbf{h}|\Omega_j)$ is the *energy function*. Using MRF modeling for appearance-based object recognition, eq (2) will become

$$j^* = \operatorname*{argmax}_j P(\mathbf{h}|\Omega_j) = \operatorname*{argmin}_j E(\mathbf{h}|\Omega_j) \qquad (4)$$

Only a few MRF approaches have been proposed for high level vision problems such as object recognition [8], due to the modeling problem for MRF on irregular sites (for a detailed discussion about this point, we refer the reader to [3]). Spin Glass-Markov Random Fields overcome this limitation and can be effectively used for robust appearance-based object recognition [3]. Next sections review SG-MRF and introduce a new energy function that allows to combine shape and color only representations in a common probabilistic framework.

## 3   Spin Glass-Markov Random Fields

Consider $k$ object classes $\Omega_1, \Omega_2, \ldots, \Omega_k$, and for each object a set of $n_j$ data samples, $d_j = \{\mathbf{x}_1^j, \ldots \mathbf{x}_{n_j}^j\}, j = 1, \ldots k$. We will suppose to extract, from each data sample $d_j$ a set of features $\{\mathbf{h}_1^j, \ldots \mathbf{h}_{n_j}^j\}$. For instance, $\mathbf{h}_{n_j}^j$ can be a color histogram computed from $\mathbf{x}_{n_j}^j$. The SG-MRF probability distribution is given by

$$P_{SGMRF}(\mathbf{h}|\Omega_j) = \frac{1}{Z} \exp\left[-E_{SGMRF}(\mathbf{h}|\Omega_j)\right], Z = \sum_{\{\mathbf{h}\}} \exp\left[-E_{SGMRF}(\mathbf{h}|\Omega_j)\right], \quad (5)$$

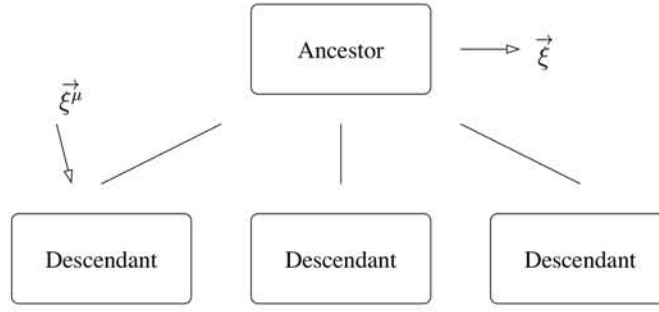

Figure 2: Hierarchical structure induced by the ultrametric energy function.

where $E_{SGMRF}(\mathbf{h}|\Omega_j)$ is a kernelized spin glass energy function. The most general SG energy is given by [1]

$$E = -\sum_{(i,j)} J_{ij}\, s_i\, s_j \qquad i,j = 1,\dots N, \tag{6}$$

where the $s_i$ are random variables taking values in $[-1,+1]$, $\mathbf{s} = (s_1,\dots,s_N)$ is a configuration and $\mathbf{J} = [\,J_{ij}\,], (i,j) = 1,\dots,N$ is the connection matrix. When the $J_{ij}$ is given by the Hopfield's prescription

$$J_{ij} = \frac{1}{N}\sum_{\mu=1}^{p} \xi_i^{(\mu)}\, \xi_j^{(\mu)}\,, \tag{7}$$

with $\{\xi^{(\mu)}\}_{\mu=1}^{p}$ given configurations of the system ( *prototypes*) having the following properties: *(a)* $\xi^{(\mu)} \perp \xi^{(\nu)}, \forall \mu \neq \nu$; *(b)* $p = \alpha N, \alpha \leq 0.14, N \to \infty$, then it can be demonstrated that $E_{SGMRF}$ becomes [3]

$$E_{SGMRF}(\mathbf{h}|\Omega_j) = -\sum_{\mu=1}^{p_j} \left[ K(\mathbf{h}, \tilde{\mathbf{h}}^{(\mu_j)}) \right]^2 , \tag{8}$$

where the function $K(\mathbf{h}, \tilde{\mathbf{h}}^{(\mu_j)})$ is a Generalized Gaussian kernel [14]:

$$K(\mathbf{x},\mathbf{y}) = \exp\{-\rho d_{a,b}(\mathbf{x},\mathbf{y})\}, \qquad d_{a,b}(\mathbf{x},\mathbf{y}) = \sum_i |x_i^a - y_i^a|^b. \tag{9}$$

$\{\tilde{\mathbf{h}}^{(\mu_j)}\}_{\mu=1}^{p_j}, j \in [1,k]$ are the prototypes selected (according to a chosen ansatz, [3]) from the training data. The number of prototypes per class must be finite, and they must satisfy the condition $K(\tilde{\mathbf{h}}^{(i)}, \tilde{\mathbf{h}}^{(l)}) = 0$, for all $i,l = 1,\dots p_j, i \neq l$ and $j = 1,\dots k$. Note that SG-MRFs are defined on features rather than on raw pixels data. The sites are fully connected, which ends in learning the neighborhood system from the training data instead of choosing it heuristically. A key characteristic of the model is that in SG-MRF the functional form of the energy is given by construction.

## 4 Ultrametric Spin Glass-Markov Random Fields

Consider the energy function (6) with the following connection matrix:

$$J_{ij} = \frac{1}{N} \sum_{\mu=1}^{p} \xi_i^{(\mu)} \xi_j^{(\mu)} \left( 1 + \sum_{\nu=1}^{q_\mu} \eta_i^{(\mu\nu)} \eta_j^{(\mu\nu)} \right) = \frac{1}{N} \sum_{\mu=1}^{p} \xi_i^{(\mu)} \xi_j^{(\mu)} + \frac{1}{N} \sum_{\mu=1}^{p} \sum_{\nu=1}^{q_\mu} \xi_i^{(\mu\nu)} \xi_j^{(\mu\nu)}$$

(10)

with $\xi_i^{(\mu\nu)} = \xi_i^{(\mu)} \eta_i^{(\mu\nu)}$. This energy induces a hierarchical organization of stored prototypes ([1], see Figure 2). The set of prototypes $\{\xi^{(\mu)}\}_{\mu=1}^{p}$ are stored at the first level of the hierarchy and are usually called the *ancestors*. Each of them will have $q$ *descendants* $\{\xi^{(\mu\nu)}\}_{\nu=1}^{q_\mu}$. The parameter $\eta_i^{(\mu\nu)}$ measures the similarity between ancestors and descendants. The first term in eq (10), right, is the Hopfield energy (6)-(7); the second is a new term that allows us to store as prototypes patterns correlated with the $\{\xi^{(\mu)}\}_{\mu=1}^{p}$; this is the case if we want to store, as separate sets of prototypes, shape only and color only representations computed from the same view. This energy will have $p + \sum_{\mu=1}^{p} q^\mu$ minima, of which $p$ absolute (ancestor level) and $\sum_{\mu=1}^{p} q^\mu$ local (descendant level). For a complete discussion on the properties of this energy, we refer the reader to [1, 4].

Here we are interested in using this energy in the SG-MRF framework shown in Section 4. To this purpose, we show that the energy (6), with the connection matrix (10), can be written as a function of scalar product between configurations [4]:

$$E = -\frac{1}{N} \sum_{ij} \left[ \frac{1}{N} \sum_{\mu=1}^{p} \xi_i^{(\mu)} \xi_j^{(\mu)} \left( 1 + \sum_{\nu=1}^{q_\mu} \eta_i^{(\mu\nu)} \eta_j^{(\mu\nu)} \right) \right] s_i s_j =$$

$$= -\left[ \frac{1}{N^2} \left[ \sum_{\mu=1}^{p} (\xi^{(\mu)} \cdot \mathbf{s})^2 + \sum_{\mu=1}^{p} \sum_{\nu=1}^{q_\mu} (\xi^{(\mu\nu)} \cdot \mathbf{s})^2 \right] \right].$$

(11)

The *ultrametric energy* (11) can be kernelized as done for the Hopfield energy and thus can be used in a MRF framework. We call the resulting new MRF model Ultrametric Spin Glass-Markov Random Fields (USG-MRF).

Now, consider the probabilistic appearance-based framework described in section 2. Given a set of data samples $d_j$ for each object class $\Omega_j, j = 1, \ldots k$, we will extract two kinds of feature vectors, $\{\mathbf{hs}_{n_j}^j\}_{j=1}^k$ containing shape information and $\{\mathbf{hc}_{n_j}^j\}_{j=1}^k$ containing color information. USG-MRF provides a straightforward manner to use the Bayes classifier (2) using both these representations separately. We will consider the color features $\{\mathbf{hc}_{n_j}^j\}_{j=1}^k$ at the ancestor level and the shape features $\{\mathbf{hs}_{n_j}^j\}_{j=1}^k$ at the descendant level. The USG-MRF energy function will be

$$E_{USGMRF} = -\sum_{\mu=1}^{p_j} [K_c(\tilde{\mathbf{hc}}^{(\mu)}, \mathbf{hc})]^2 - \sum_{\mu=1}^{p_j} \sum_{\nu=1}^{q_\mu} [K_s(\tilde{\mathbf{hs}}^{(\mu\nu)}, \mathbf{hs})]^2,$$

(12)

where $\{\tilde{\mathbf{hc}}^{(\mu)}\}_{\mu=1}^{p_j}$ will be the set of prototypes relative to the ancestor level, and $\{\tilde{\mathbf{hs}}^{(\mu\nu)}\}_{\nu=1}^{q_\mu}, \mu = 1, \ldots p_j$ the set of prototypes at the descendant level. These prototypes are selected from the training data as described in section 3 for SG-MRF. $K_c$ is the generalized Gaussian kernel at the ancestor level, and $K_s$ is the generalized Gaussian kernel at the descendant level. We stress that the kernel must

be the same at each level of the hierarchy, but can be different between levels (as to say between ancestor and descendant). The Bayes classifier based on USG-MRF will be

$$j^* = \underset{j}{\operatorname{argmin}} \left\{ -\sum_{\mu=1}^{p_j} [K_c(\tilde{\mathbf{hc}}^{(\mu)}, \mathbf{hc})]^2 - \sum_{\mu=1}^{p_j} \sum_{\nu=1}^{q_\mu} [K_s(\tilde{\mathbf{hs}}^{(\mu\nu)}, \mathbf{hs})]^2 \right\}. \quad (13)$$

Note that the parametric form of kernels is known (eq (9); thus, when (U)SG-MRF is used in a Bayes classifier for classification purposes, it permits to **learn** the kernel to be used from the training data, with a leave-one-out strategy.

## 5 Experiments

In order to show the effectiveness of USG-MRF for appearance-based object recognition, we perform several sets of experiments. All of them were ran on the COIL database [9]; it consists of 7200 color images of 100 objects (72 views for object); each image is of $128 \times 128$ pixels. The images were obtained by placing the objects on a turntable and taking a view every $5°$. In all the experiments we performed, the training set consisted of 12 views per object (one every $30°$). The remaining views constituted the test set.

Among the many representations proposed in literature, we chose a shape only and color only representation, and we ran experiments using these representations separated, concatenated together in a common feature vector and combined together in the USG-MRF. The purpose of these experiments is to prove the effectiveness of the USG-MRF model rather than select the optimal combination for the shape and color representations. Thus, we limited the experiments to one shape only and one color only representations; but USG-MRF can be applied to any other kind of shape and/or color representation (see for instance [4]).

As color only representation, we chose two dimensional $rg$ Color Histogram (CH), with resolution of bin axis equal to 8 [13]. The CH was normalized to 1. As shape only representation, we chose Multidimensional receptive Field Histograms (MFH) [11], with two local characteristics based on Gaussian derivatives along $x$ and $y$ directions, with $\sigma = 1.0$ and resolution of bin axis equal to 8. The histograms were normalized to 1. These two representations were used for performing the following sets of experiments:
• **Shape experiments**: we ran the experiments using the shape features only. Classification was performed using SG-MRF with the kernelized Hopfield energy (6)-(7). The kernel parameters $(a, b, \rho)$ were learned using a leave-one-out strategy. The results were benchmarked with those obtained with a $\chi^2$ and $\cap$ similarity measures, which proved to be very effective for this representation, and with SVM with Gaussian kernel, $\rho \in [0.001, 10]$ (here we report only the best results obtained).
• **Color experiments**: we ran the experiments using the color features only. Classification and benchmarking were performed as in the shape experiment.
• **Color-Shape experiments**: we ran the experiments using the color and shape features concatenated together to form a unique feature vector. Again, classification and benchmarking were performed as in the shape experiment.
• **Ultrametric experiment**: we ran a single experiment using the shape and color representation disjoint in the USG-MRF framework. The kernel parameters relative

to each level ($a_s, b_s, \rho_s$ and $a_c, b_c, \rho_c$) are learned with the leave-one-out technique. Results obtained with this approach cannot be directly benchmarked with other similarity measures. Anyway, it is possible to compare the obtained results with those of the previous experiments.

Table 1 reports the error rates obtained for the 4 sets of experiments.

|  | Color (%) | Shape (%) | Color-Shape (%) | Ultrametric (%) |
|---|---|---|---|---|
| $\chi^2$ | 23.47 | 9.47 | 19.17 | |
| $\cap$ | 25.68 | 24.94 | 21.72 | |
| SVM | 19.78 | 25.3 | 18.38 | |
| SG-MRF | **20.10** | **6.28** | **8.43** | **3.55** |

Table 1: Classification results; we report for each set of experiments the obtained error rates.

Results presented in Table 1 show that for all series of experiments, for all representations, SG-MRF always gave the best recognition result. Moreover, the overall best recognition result is obtained with USG-MRF. USG-MRF has an increase of performance of +2.73% with respect to SG-MRF, best result, and of +5.92% with respect to $\chi^2$ (best result obtained with a non SG-MRF technique). Table 2 shows some examples of objects misclassified by SG-MRF and correctly classified by USG-MRF. We see that USG-MRF classifies correctly in cases where shape only or color only gives the right answer (but not both, and not in the concatenated representation; Table 2, left and middle column), and also in cases where color only and shape only don't classify correctly (Table 2, right column). These examples show clearly that the better performance of USG-MRF is due to its hierarchical structure that permits to use different kernels on different features, thus to weight their relevance in a flexible manner with respect to the considered application.

We remark once again that all the kernel parameters (thus ultimately the kernel itself) are **learned** from the training data; to the best of our knowledge (U)SG-MRF is the first kernel method for vision application that doesn't select heuristically the kernel to be used.

| | | 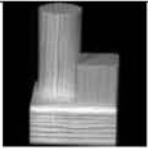 |  |  |
|---|---|---|---|---|
| USG-MRF | | **1st match** | **1st match** | **1st match** |
| $\text{SG} - \text{MRF}_s$ | | 2nd match | **1st match** | 3rd match |
| $\text{SG} - \text{MRF}_c$ | | **1st match** | 2nd match | 7th match |
| $\text{SG} - \text{MRF}_{sc}$ | | 3rd match | 2nd match | 5th match |

Table 2: Classification results for sample objects; USG-MRF classifies always correctly even when color only ($\text{SG} - \text{MRF}_s$), color only ($\text{SG} - \text{MRF}_c$) and common representation ($\text{SG} - \text{MRF}_{sc}$) fail (right column).

## 6 Summary

In this paper we presented a kernel method that permits us to combine color and shape information for appearance-based object recognition. It does not require us to define a new common representation, but use the power of kernels to combine different representations together in an effective manner. This result is achieved using results of statistical mechanics of Spin Glasses combined with Markov Random Fields via kernel functions. Experiments confirm the effectiveness of the proposed approach. Future work will explore the possibility to use different representations for color and shape and to use this method for tackling other challenging problems in object recognition, such as recognition of objects in heterogeneous background and under different lighting conditions.

**Acknowledgments**

This work has been supported by the "Graduate Research Center of the University of Erlangen-Nuremberg for 3D Image Analysis and Synthesis", and by the Foundation BLANCEFLOR Boncompagni-Ludovisi.

## References

[1] D. J. Amit, *"Modeling Brain Function"*, Cambridge University Press, 1989.

[2] S. Belongie, J. Malik, J. Puzicha, "Matching Shapes", *ICCV01*, 454-461.

[3] B. Caputo, S. Bouattour, H. Niemann, "A new kernel method for robust appearance-based object recognition: Spin Glass-Markov random fields", submitted to *PR*, available at $http : //www.ski.org/ALYuille_lab/$.

[4] B. Caputo, Gy. Dorko, H. Niemann, "An ultrametric approach to object recognition", submitted to *VMV02*, availabe at $http : //www.ski.org/ALYuille_lab/$.

[5] A. Leonardis, H. Bischof, "Robust recognition using eigenimages", CVIU,78:99-118, 2000.

[6] J. Matas, R, Marik, J. Kittler, "On representation and matching of multi-coloured objects", *Proc ICCV95*, 726-732, 1995.

[7] B. W. Mel, "SEEMORE: combining color, shape and texture histogramming in a neurally-inspired approach to visual object recognition", *NC*, 9: 777-804, 1997

[8] J.W. Modestino, J. Zhang. "A Markov random field model–based approach to image interpretation". *PAMI*, 14(6),606–615,1992.

[9] Nene, S. A., Nayar, S. K., Murase, H., "Columbia Object Image Library (COIL-100)", *TR* CUCS-006-96, Dept. Comp. Sc., Columbia University, 1996.

[10] Pontil, M., Verri, A. "Support Vector Machines for 3D Object Recognition", *PAMI*, 20(6):637-646, 1998.

[11] B. Schiele, J. L. Crowley, "Recognition without correspondence using multidimensional receptive field histograms", *IJCV*, 36(1),:31- 52, 2000.

[12] D. Slater, G. Healey, "Combining color and geometric information for the illumination invariant recognition of 3-D objects", *Proc ICCV95*, 563-568, 1995.

[13] M. Swain, D. Ballard, "Color indexing",*IJCV*, 7(1):11-32, 1991.

[14] B. Schölkopf, A. J. Smola, *Learning with kernels*, 2002, the MIT Press, Cambridge, MA.
